# Artefactual Structure from Least Squares Multidimensional Scaling

**Nicholas P. Hughes**
Department of Engineering Science
University of Oxford
Oxford, 0X1 3PJ, UK
*nph@robots.ox.ac.uk*

**David Lowe**
Neural Computing Research Group
Aston University
Birmingham, B4 7ET, UK
*d.lowe@aston.ac.uk*

## Abstract

We consider the problem of illusory or artefactual structure from the visualisation of high-dimensional structureless data. In particular we examine the role of the distance metric in the use of topographic mappings based on the statistical field of multidimensional scaling. We show that the use of a squared Euclidean metric (i.e. the SSTRESS measure) gives rise to an annular structure when the input data is drawn from a high-dimensional isotropic distribution, and we provide a theoretical justification for this observation.

## 1 Introduction

The discovery of meaningful patterns and relationships from large amounts of multivariate data is a significant and challenging problem with close ties to the fields of pattern recognition and machine learning, and important applications in the areas of data mining and knowledge discovery in databases (KDD).

For many real-world high-dimensional data sets (such as collections of images, or multichannel recordings of biomedical signals) there will generally be strong correlations between neighbouring observations, and thus we expect that the data will lie on a lower dimensional (possibly nonlinear) manifold embedded in the original data space. One approach to the aforementioned problem then is to find a *faithful*[1] representation of the data in a lower dimensional space. Typically this space is chosen to be two- or three-dimensional, thus facilitating the visualisation and exploratory analysis of the intrinsic low-dimensional structure in the data (which would otherwise be masked by the dimensionality of the data space).

In this context then, an effective dimensionality reduction algorithm should seek to extract the underlying relationships in the data with minimum loss of information. Conversely, any interesting patterns which are present in the visualisation space should be representative of similar patterns in the original data space, and not *artefacts* of the dimensionality reduction process.

Although much effort has been focused on the former problem of optimal structure elucidation (see [7, 10] for recent approaches to dimensionality reduction), comparatively little work has been undertaken on the latter (and equally important) problem of artefactual structure. This shortcoming was recently highlighted in a controversial example of the application of visualisation techniques to neuroanatomical connectivity data derived from the primate visual cortex [12, 9, 13, 3].

In this paper we attempt to redress the balance by considering the visualisation of high-dimensional *structureless* data through the use of topographic mappings based on the statistical field of multidimensional scaling (MDS). This is an important class of mappings which have recently been brought into the neural network domain [5], and have significant connections to modern kernel-based algorithms such as kernel PCA [11].

The organisation of the remainder of this paper is as follows: In section 2 we introduce the technique of multidimensional scaling and relate this to the field of topographic mappings. In section 3 we show how under certain conditions such mappings can give rise to artefactual structure. A theoretical analysis of this effect is then presented in section 4.

## 2   Multidimensional Scaling and Topographic Mappings

The visualisation of experimental data which is characterised by pairwise proximity values is a common problem in areas such as psychology, molecular biology and linguistics. Multidimensional scaling (MDS) is a statistical technique which can be used to construct a spatial configuration of $n$ points in a (typically) two- or three-dimensional space given a matrix of pairwise proximity values between $n$ objects. The proximity matrix provides a measure of the *similarity* or *dissimilarity* between the objects, and the geometric layout of the resulting MDS configuration reflects the relationships between the objects as defined by this matrix. In this way the information contained within the proximity matrix can be captured by a more succinct *spatial model* which aids visualisation of the data and improves understanding of the processes that generated it.

In many situations, the raw dissimilarities will not be representative of actual inter-point distances between the objects, and thus will not be suitable for embedding in a low-dimensional space. In this case the dissimilarities can be transformed into a set of values more suitable for embedding through the use of an appropriate transformation:

$$\hat{\delta}_{ij} = f(\delta_{ij})$$

where $f$ represents the transformation function and $\hat{\delta}_{ij}$ are the resulting transformed dissimilarities (which are termed "disparities"). The aim of metric MDS then is that the transformed dissimilarities $\hat{\delta}_{ij}$ should correspond as closely as possible to the inter-point distances $d_{ij}$ in the resulting configuration[2].

Metric MDS can be formulated as a continuous optimisation problem through the definition of an appropriate error function. In particular, *least squares* scaling algorithms directly seek to minimise the sum-of-squares error between the disparities and the inter-point distances. This error, or STRESS[3] measure, is given by:

$$\text{STRESS} = \frac{1}{\sum_{i,j} \hat{\delta}_{ij}^2} \sum_i \sum_{j>i} w_{ij} \left( \hat{\delta}_{ij} - d_{ij} \right)^2 \qquad (1)$$

where the term $1/\sum_{i,j}\hat{\delta}_{ij}^2$ is a normalising constant which reduces the sensitivity of the measure to the number of points and the scaling of the disparities, and the $w_{ij}$ are the weighting factors. It is straightforward to differentiate this STRESS measure with respect to the configuration points $\mathbf{y}_i$ and minimise the error through the use of standard nonlinear optimisation techniques.

An alternative and commonly used error function, which is referred to as SSTRESS, is given by:

$$\text{SSTRESS} = \frac{1}{\sum_{i,j}\hat{\delta}_{ij}^2} \sum_i \sum_{j>i} \left(\hat{\delta}_{ij}^2 - d_{ij}^2\right)^2 \tag{2}$$

which represents the sum-of-squares error between *squared* disparities and *squared* distances. The primary advantage of the SSTRESS measure is that it can be efficiently minimised through the use of an alternating least squares procedure[4] [1].

Closely related to the field of Metric MDS is Sammon's mapping [8], which takes as its input a set of *high-dimensional vectors* and seeks to produce a set of lower dimensional vectors such that the following error measure is minimised:

$$E_{sammon} = \frac{1}{\sum_{i,j}d_{ij}^*} \sum_i \sum_{j>i} \frac{(d_{ij}^* - d_{ij})^2}{d_{ij}^*} \tag{3}$$

where the $d_{ij}^*$ are the inter-point Euclidean distances in the data space: $d_{ij}^* = \|\mathbf{x}_i - \mathbf{x}_j\|$, and the $d_{ij}$ are the corresponding inter-point Euclidean distances in the feature or map space: $d_{ij} = \|\mathbf{y}_i - \mathbf{y}_j\|$.

Ignoring the normalising constant, Sammon's mapping is thus equivalent to least squares metric MDS with the disparities taken to be the raw inter-point distances in the data space and the weighting factors given by $w_{ij} = 1/d_{ij}^*$. Lowe (1993) termed such a mapping based on the minimisation of an error measure of the form $\sum_{i,j}(d_{ij}^* - d_{ij})^2$ a *topographic mapping*, since this constraint "optimally preserves the geometric structure in the data" [5].

Interestingly the choice of the STRESS or SSTRESS measure in MDS has a more natural interpretation when viewed within the framework of Sammon's mapping. In particular, STRESS corresponds to the use of the standard Euclidean distance metric whereas SSTRESS corresponds to the use of the *squared* Euclidean distance metric. In the next section we show that this choice of metric can lead to markedly different results when the input data is sampled from a high-dimensional isotropic distribution.

## 3  Emergence of Artefactual Structure

In order to investigate the problem of artefactual structure we consider the visualisation of high-dimensional structureless data (where we use the term "structureless" to indicate that the data density is equal in all directions from the mean and varies only gradually in any direction). Such data can be generated by sampling from an *isotropic* distribution (such as a spherical Gaussian), which is characterised by a covariance matrix that is proportional to the identity matrix, and a skewness of zero.

We created four structureless data sets by randomly sampling 1000 i.i.d. points from unit hypercubes of dimensions $p = 5$, 10, 30 and 100. For each data set, we generated a pair

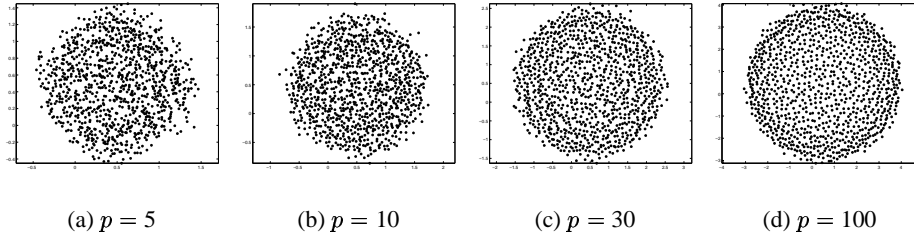

(a) $p = 5$      (b) $p = 10$      (c) $p = 30$      (d) $p = 100$

Figure 1: Final map configurations produced by STRESS mappings of data uniformly randomly distributed in unit hypercubes of dimension $p$.

of 2-D configurations by minimising[5] STRESS and SSTRESS error measures of the form $\sum_{i,j}(d_{ij}^* - d_{ij})^2$ and $\sum_{i,j}(d_{ij}^{*2} - d_{ij}^2)^2$ respectively. The process was repeated fifty times (for each individual error function and data set) using different initial configurations of the map points, and the configuration with the lowest final error was retained.

As previously noted, the choice of the STRESS or SSTRESS error measure is best viewed as a choice of distance metric, where STRESS corresponds to the standard Euclidean metric and SSTRESS corresponds to the *squared* Euclidean metric. Figure 1 shows the resulting configurations from the STRESS mappings. It is clear that each configuration has captured the isotropic nature of the associated data set, and there are no spurious patterns or clusters evident in the final visualisation plots.

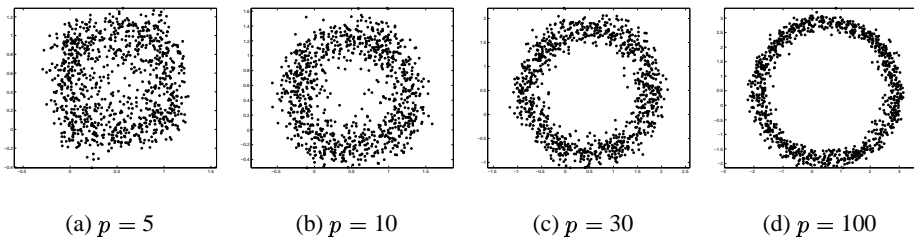

(a) $p = 5$      (b) $p = 10$      (c) $p = 30$      (d) $p = 100$

Figure 2: Final map configurations produced by SSTRESS mappings of data uniformly randomly distributed in unit hypercubes of dimension $p$.

Figure 2 shows the resulting configurations from the SSTRESS mappings. The configurations exhibit significant artefactual structure, which is characterised by a tendency for the map points to cluster in a circular fashion. Furthermore, the degree of clustering increases with increasing dimensionality of the data space $p$ (and is clearly evident for $p$ as low as 10).

Although the tendency for SSTRESS configurations to cluster in a circular fashion has been noted in the MDS literature [2], the connection between artefactual structure and the choice of distance metric has not been made. Indeed, in the next section we show analytically that the use of the squared Euclidean metric leads to a globally optimal solution corresponding to an annular structure.

To date, the most significant work on this problem is that of Klock and Buhmann [4], who proposed a novel transformation of the dissimilarities (i.e. the squared inter-point distances

in the data space) such that "the final disparities are more suitable for Euclidean embedding". However this transformation assumes that the input data are drawn from a spherical Gaussian distribution[6], which is inappropriate for most real-world data sets of interest.

## 4   Theoretical Analysis of Artefactual Structure

In this section we present a theoretical analysis of the artefactual structure problem. A $q$ dimensional map configuration is considered to be the result of a SSTRESS mapping of a data set of $N$ i.i.d. points drawn from a $p$ dimensional isotropic distribution (where $p \gg q$). The set of data points is given by the $N$ x $p$ matrix $\mathbf{X} = (\mathbf{x}_1, \mathbf{x}_2, \dots, \mathbf{x}_N)^{\mathrm{T}}$ and similarly the set of map points is given by the $N$ x $q$ matrix $\mathbf{Y} = (\mathbf{y}_1, \mathbf{y}_2, \dots, \mathbf{y}_N)^{\mathrm{T}}$.

We begin by defining the derivative of the SSTRESS error measure $E = \sum_{i,j}(d_{ij}^{*2} - d_{ij}^2)^2$ with respect to a particular map vector $\mathbf{y}_i$:

$$\frac{\partial E}{\partial \mathbf{y}_i} = -4 \sum_{j \neq i}(d_{ij}^{*2} - d_{ij}^2)\,(\mathbf{y}_i - \mathbf{y}_j) \tag{4}$$

The inter-point distances $d_{ij}^{*2}$ and $d_{ij}^2$ are given by:

$$d_{ij}^{*2} = \|\mathbf{x}_i - \mathbf{x}_j\|^2 = (\mathbf{x}_i - \mathbf{x}_j)^{\mathrm{T}}(\mathbf{x}_i - \mathbf{x}_j) = \mathbf{x}_i^{\mathrm{T}}\mathbf{x}_i + \mathbf{x}_j^{\mathrm{T}}\mathbf{x}_j - 2\,\mathbf{x}_i^{\mathrm{T}}\mathbf{x}_j$$
$$d_{ij}^2 = \|\mathbf{y}_i - \mathbf{y}_j\|^2 = (\mathbf{y}_i - \mathbf{y}_j)^{\mathrm{T}}(\mathbf{y}_i - \mathbf{y}_j) = \mathbf{y}_i^{\mathrm{T}}\mathbf{y}_i + \mathbf{y}_j^{\mathrm{T}}\mathbf{y}_j - 2\,\mathbf{y}_i^{\mathrm{T}}\mathbf{y}_j$$

Equation (4) can therefore be expanded to:

$$\frac{\partial E}{\partial \mathbf{y}_i} = -4\sum_{j \neq i}(\mathbf{x}_i^{\mathrm{T}}\mathbf{x}_i - \mathbf{y}_i^{\mathrm{T}}\mathbf{y}_i)\,(\mathbf{y}_i - \mathbf{y}_j) - 4\sum_{j \neq i}(\mathbf{x}_j^{\mathrm{T}}\mathbf{x}_j - \mathbf{y}_j^{\mathrm{T}}\mathbf{y}_j)\,(\mathbf{y}_i - \mathbf{y}_j)$$
$$-8\sum_{j \neq i}(\mathbf{y}_i^{\mathrm{T}}\mathbf{y}_j)\,\mathbf{y}_i + 8\sum_{j \neq i}(\mathbf{x}_i^{\mathrm{T}}\mathbf{x}_j)\,\mathbf{y}_i + 8\sum_{j \neq i}(\mathbf{y}_i^{\mathrm{T}}\mathbf{y}_j)\,\mathbf{y}_j - 8\sum_{j \neq i}(\mathbf{x}_i^{\mathrm{T}}\mathbf{x}_j)\,\mathbf{y}_j$$

We can immediately simplify some of these terms as follows:

$$\sum_{j \neq i}(\mathbf{y}_i^{\mathrm{T}}\mathbf{y}_j)\,\mathbf{y}_i = \sum_{j \neq i}\mathbf{y}_i\,(\mathbf{y}_i^{\mathrm{T}}\mathbf{y}_j) = \mathbf{y}_i\mathbf{y}_i^{\mathrm{T}}\sum_{j \neq i}\mathbf{y}_j$$
$$\sum_{j \neq i}(\mathbf{x}_i^{\mathrm{T}}\mathbf{x}_j)\,\mathbf{y}_i = \sum_{j \neq i}\mathbf{y}_i\,(\mathbf{x}_i^{\mathrm{T}}\mathbf{x}_j) = \mathbf{y}_i\mathbf{x}_i^{\mathrm{T}}\sum_{j \neq i}\mathbf{x}_j$$
$$\sum_{j \neq i}(\mathbf{y}_i^{\mathrm{T}}\mathbf{y}_j)\,\mathbf{y}_j = \sum_{j \neq i}\mathbf{y}_j\,(\mathbf{y}_j^{\mathrm{T}}\mathbf{y}_i) = \sum_{j \neq i}(\mathbf{y}_j\mathbf{y}_j^{\mathrm{T}})\,\mathbf{y}_i$$
$$\sum_{j \neq i}(\mathbf{x}_i^{\mathrm{T}}\mathbf{x}_j)\,\mathbf{y}_j = \sum_{j \neq i}\mathbf{y}_j\,(\mathbf{x}_j^{\mathrm{T}}\mathbf{x}_i) = \sum_{j \neq i}(\mathbf{y}_j\mathbf{x}_j^{\mathrm{T}})\,\mathbf{x}_i$$

Thus at a stationary point of the error (i.e. $\frac{\partial E}{\partial \mathbf{y}_i} = \mathbf{0}$), we have:

$$(\mathbf{x}_i^{\mathrm{T}}\mathbf{x}_i - \mathbf{y}_i^{\mathrm{T}}\mathbf{y}_i)\left(\mathbf{y}_i - \frac{1}{N-1}\sum_{j \neq i}\mathbf{y}_j\right) + \frac{1}{N-1}\sum_{j \neq i}(\mathbf{x}_j^{\mathrm{T}}\mathbf{x}_j - \mathbf{y}_j^{\mathrm{T}}\mathbf{y}_j)\,(\mathbf{y}_i - \mathbf{y}_j)$$

$$= \frac{2}{N-1} \left( \mathbf{y}_i \mathbf{x}_i^\mathsf{T} \sum_{j\neq i} \mathbf{x}_j - \mathbf{y}_i \mathbf{y}_i^\mathsf{T} \sum_{j\neq i} \mathbf{y}_j + \sum_{j\neq i} \left( \mathbf{y}_j \mathbf{y}_j^\mathsf{T} \right) \mathbf{y}_i - \sum_{j\neq i} \left( \mathbf{y}_j \mathbf{x}_j^\mathsf{T} \right) \mathbf{x}_i \right) \quad (5)$$

Since the error $E$ is a function of the inter-point distances only, we can centre both the data points and the map points on the origin without loss of generality. For large $N$ we have:

$$\frac{1}{N-1} \sum_{j\neq i} \mathbf{y}_j \approx \mathbf{0}_{q,1} \qquad\qquad \frac{1}{N-1} \sum_{j\neq i} \mathbf{x}_j \approx \mathbf{0}_{p,1}$$

$$\frac{1}{N-1} \sum_{j\neq i} \mathbf{y}_j \mathbf{y}_j^\mathsf{T} \approx \Sigma_\mathbf{Y} \qquad\qquad \frac{1}{N-1} \sum_{j\neq i} \mathbf{y}_j \mathbf{x}_j^\mathsf{T} \approx \Sigma_\mathbf{YX}$$

$$\frac{1}{N-1} \sum_{j\neq i} \mathbf{y}_j^\mathsf{T} \mathbf{y}_j \approx \mathrm{tr}\{\Sigma_\mathbf{Y}\} \qquad \frac{1}{N-1} \sum_{j\neq i} \mathbf{x}_j^\mathsf{T} \mathbf{x}_j \approx \mathrm{tr}\{\Sigma_\mathbf{x}\}$$

where $\mathbf{0}_{m,n}$ is the $m$ x $n$ zero matrix, $\Sigma_\mathbf{Y}$ is the covariance matrix of the map vectors, $\Sigma_\mathbf{YX}$ is the covariance matrix of the map vectors and the data vectors, and $\mathrm{tr}\{\cdot\}$ is the matrix trace operator.

Thus equation (5) reduces to:

$$\left( \mathbf{x}_i^\mathsf{T} \mathbf{x}_i - \mathbf{y}_i^\mathsf{T} \mathbf{y}_i \right) \mathbf{y}_i + \frac{1}{N-1} \sum_{j\neq i} \left( \mathbf{y}_j^\mathsf{T} \mathbf{y}_j \right) \mathbf{y}_j - \frac{1}{N-1} \sum_{j\neq i} \left( \mathbf{x}_j^\mathsf{T} \mathbf{x}_j \right) \mathbf{y}_j$$

$$= 2 \Sigma_\mathbf{Y} \mathbf{y}_i - 2 \Sigma_\mathbf{YX} \mathbf{x}_i + \left( \mathrm{tr}\{\Sigma_\mathbf{Y}\} - \mathrm{tr}\{\Sigma_\mathbf{x}\} \right) \mathbf{y}_i \quad (6)$$

This represents a general expression for the value of the map vector $\mathbf{y}_i$ at a stationary point of the SSTRESS error, *regardless* of the nature of the input data distribution. However we are interested in the case where the input data is drawn from a high-dimensional *isotropic* distribution.

If the data space is isotropic then a stationary point of the error will correspond to a similarly isotropic map space[7]. Thus, at a stationary point, we have for large $N$:

$$\Sigma_\mathbf{Y} \approx \sigma_y^2 \mathbf{I}_q$$

$$\Sigma_\mathbf{YX} \approx \mathbf{0}_{q,p}$$

$$\mathrm{tr}\{\Sigma_\mathbf{Y}\} - \mathrm{tr}\{\Sigma_\mathbf{x}\} \approx q\,\sigma_y^2 - p\,\sigma_x^2$$

where $\mathbf{I}_q$ is the $q$ x $q$ identity matrix, and $\sigma_y^2$ and $\sigma_x^2$ are the variances in the map space and the data space respectively.

Finally, consider the expression:

$$\frac{1}{N-1} \sum_{j\neq i} \left( \mathbf{y}_j^\mathsf{T} \mathbf{y}_j \right) \mathbf{y}_j - \frac{1}{N-1} \sum_{j\neq i} \left( \mathbf{x}_j^\mathsf{T} \mathbf{x}_j \right) \mathbf{y}_j$$

The first term is the third order moment, which is zero for an isotropic distribution [6]. For high-dimensional data (i.e. large $p$) the second term can be simplified to:

$$\frac{1}{N-1} \sum_{j\neq i} \left( \mathbf{x}_j^\mathsf{T} \mathbf{x}_j \right) \mathbf{y}_j = \frac{p}{N-1} \sum_{j\neq i} \left( \frac{1}{p} \sum_{k=1}^{p} x_{jk}^2 \right) \mathbf{y}_j \approx \frac{p\,\sigma_x^2}{N-1} \sum_{j\neq i} \mathbf{y}_j \approx \mathbf{0}_{q,1} \quad (7)$$

Thus the equation governing the stationary points of the SSTRESS error is given by:

$$\left(\mathbf{x}_i^{\mathsf{T}}\mathbf{x}_i - \mathbf{y}_i^{\mathsf{T}}\mathbf{y}_i + p\,\sigma_x^2 - (q+2)\,\sigma_y^2\right)\mathbf{y}_i = \mathbf{0}_{q,1}$$

At the minimum error configuration, we have:

$$\mathbf{x}_i^{\mathsf{T}}\mathbf{x}_i - \mathbf{y}_i^{\mathsf{T}}\mathbf{y}_i + p\,\sigma_x^2 - (q+2)\,\sigma_y^2 = 0$$

Summing over all points $i$, gives:

$$\sum_{i=1}^{N}\left(\mathbf{x}_i^{\mathsf{T}}\mathbf{x}_i - \mathbf{y}_i^{\mathsf{T}}\mathbf{y}_i + p\,\sigma_x^2 - (q+2)\,\sigma_y^2\right) = 0$$

$$\therefore \quad \frac{1}{N}\sum_{i=1}^{N}\mathbf{x}_i^{\mathsf{T}}\mathbf{x}_i - \frac{1}{N}\sum_{i=1}^{N}\mathbf{y}_i^{\mathsf{T}}\mathbf{y}_i + p\,\sigma_x^2 - (q+2)\,\sigma_y^2 = 0$$

$$\therefore \quad \mathrm{tr}\{\Sigma_{\mathbf{x}}\} - \mathrm{tr}\{\Sigma_{\mathbf{Y}}\} + p\,\sigma_x^2 - (q+2)\,\sigma_y^2 = 0$$

$$\therefore \quad \sigma_y^2 = \frac{p}{q+1}\,\sigma_x^2 \tag{8}$$

Thus, for large $p$, the variance of the map points is related to the variance of the data points by a factor of $\frac{p}{q+1}$. Table 1 shows the values of the observed and predicted map variances for 1000 data points sampled randomly from uniform distributions in the interval $[0,1]^p$ (i.e. $\sigma_x^2 = 0.083$) of dimensions $p = 5, 10, 30$, and $100$. Clearly as the dimension of the data space $p$ increases, so too does the accuracy of the approximation given by equation (7), and therefore the accuracy of equation (8).

| Dimension $p$ | Number of points $N$ | $\sigma_y^2$ observed | $\sigma_y^2$ predicted | Percentage error |
|---|---|---|---|---|
| 5 | 1000 | 0.166 | 0.139 | 16.4% |
| 10 | 1000 | 0.303 | 0.278 | 8.1% |
| 30 | 1000 | 0.864 | 0.835 | 3.4% |
| 100 | 1000 | 2.823 | 2.783 | 1.4% |

Table 1: A comparison of the predicted and observed map variances.

We can show that this mismatch in variances in the two spaces results in the map points clustering in a circular fashion by considering the expected squared distance of the map points from the origin (i.e. the expected *squared radius* $R^2$ of the annulus):

$$\mathcal{E}\{R^2\} = \frac{1}{N}\sum_{i=1}^{N}\mathbf{y}_i^{\mathsf{T}}\mathbf{y}_i = q\,\sigma_y^2 = \frac{p\,q}{q+1}\,\sigma_x^2 \tag{9}$$

In addition we can derive an analytic expression for $\mathcal{E}\{R^4\}$. For simplicity, consider a two-dimensional map space $\mathbf{y} = (y_1, y_2)^{\mathsf{T}}$. Then we have:

$$\mathcal{E}\{R^4\} = \mathcal{E}\{y_1^4 + 2y_1^2 y_2^2 + y_2^4\}$$

$$= \mathcal{E}\{y_1^4\} + 2\,\mathcal{E}\{y_1^2\}\mathcal{E}\{y_2^2\} + \mathcal{E}\{y_2^4\}$$

$$= 4\,\sigma_y^4 \tag{10}$$

where the expectation over $y_1^2 y_2^2$ separates since $y_1^2$ and $y_2^2$ will be uncorrelated due to the isotropic nature of $\mathbf{y}$. In general for a $q$-dimensional map space we have that $\mathcal{E}\{R^4\} = q^2\,\sigma_y^4$. Thus the *variance* of $R^2$ is given by:

$$Var(R^2) = \mathcal{E}\{R^4\} - (\mathcal{E}\{R^2\})^2 = 0$$

Hence for large $p$ the optimal configuration will be an annulus or ring shape, as observed in figure 2.

## 5   Conclusions

We have investigated the problem or artefactual or illusory structure from topographic mappings based upon least squares scaling algorithms from multidimensional scaling. In particular we have shown that the use of a squared Euclidean distance metric (i.e. the SSTRESS measure) gives rise to an annular structure when the input data is drawn from a high-dimensional isotropic distribution. A theoretical analysis of this problem was presented and a simple relationship between the variance of the map and the data points was derived. Finally we showed that this relationship results in an optimal configuration which is characterised by the map points clustering in a circular fashion.

### Acknowledgments

We thank Miguel Carreira-Perpiñán for useful comments on this work.

## Footnotes

[1]By "faithful" we mean that the underlying geometric structure in the data space, which characterises the informative relationships in the data, is preserved in the visualisation space.

[2]This is in contrast to nonmetric MDS which requires that only the *ordering* of the disparities corresponds to the ordering of the inter-point distances (and thus that the disparities are some arbitrary monotonically increasing function of the distances).

[3]STRESS is an acronym for STandard REsidual Sum of Squares.

[4] The SSTRESS measure now forms the basis of the ALSCAL implementation of MDS, which is included as part of the SPSS software package for statistical data analysis.

[5]We used a conjugate gradients optimisation algorithm.

[6]In this case the squared inter-point distances will follow a $\chi^2$ distribution.

[7]This is true regardless of the initial distribution of the map points, although a highly non-uniform initial configuration would take significantly longer to reach a local minimum of the error function.

## References

[1]  T. F. Cox and M. A. A. Cox. *Multidimensional scaling*. Chapman and Hall, London, 1994.

[2]  J. deLeeuw and B. Bettonvil. An upper bound for sstress. *Psychometrika*, 51:149 – 153, 1986.

[3]  G. J. Goodhill, M. W. Simmen, and D. J. Willshaw. An evaluation of the use of multidimensional scaling for understanding brain connectivity. *Philosophical Transactions of the Royal Society, Series B*, 348:256 – 280, 1995.

[4]  H. Klock and J. M. Buhmann.  Multidimensional scaling by deterministic annealing.  In M. Pelillo and E. R. Hancock, editors, *Energy Minimization Methods in Computer Vision and Pattern Recognition*, Proc. Int. Workshop EMMCVPR '97, Venice, Italy, pages 246–260. Springer Lecture Notes in Computer Science, 1997.

[5]  D. Lowe and M. E. Tipping. Neuroscale: Novel topographic feature extraction with radial basis function networks. In M. C. Mozer, M. I. Jordan, and T. Petsche, editors, *Advances in Neural Information Processing Systems 9*. Cambridge, MA: MIT Press, 1997.

[6]  K. V. Mardia, J. T. Kent, and J. M. Bibby. *Multivariate analysis*. Academic Press, 1997.

[7]  S. T. Roweis, L. K. Saul, and G. E. Hinton. Global coordination of local linear models. In T. G. Dietterich, S. Becker, and Z. Ghahramani, editors, *Advances in Neural Information Processing Systems 14*. Cambridge, MA: MIT Press, 2002.

[8]  J. W. Sammon. A nonlinear mapping for data structure analysis. *IEEE Transactions On Computers*, C-18(5):401 – 409, 1969.

[9]  M. W. Simmen, G. J. Goodhill, and D. J. Willshaw.  Scaling and brain connectivity. *Nature*, 369:448–450, 1994.

[10] J. B. Tenenbaum. Mapping a manifold of perceptual observations. In M. I. Jordan, M. J. Kearns, and S. A. Solla, editors, *Advances in Neural Information Processing Systems 10*. Cambridge, MA: MIT Press, 1998.

[11] C. K. Williams. On a connection between kernel PCA and metric multidimensional scaling. In T. K. Leen, T. G. Diettrich, and V. Tresp, editors, *Advances in Neural Information Processing Systems 13*. Cambridge, MA: MIT Press, 2001.

[12] M. P. Young.  Objective analysis of the topological organization of the primate cortical visual system. *Nature*, 358:152–155, 1992.

[13] M. P. Young, J. W. Scannell, M. A. O'Neill, C. C. Hilgetag, G. Burns, and C. Blakemore. Non-metric multidimensional scaling in the analysis of neuroanatomical connection data and the organization of the primate cortical visual system. *Philosophical Transactions of the Royal Society, Series B*, 348:281 – 308, 1995.
